# On the Relation Between Low Density Separation, Spectral Clustering and Graph Cuts

**Hariharan Narayanan**
Department of Computer Science
University of Chicago
Chicago IL 60637
hari@cs.uchicago.edu

**Mikhail Belkin**
Department of Computer Science and Engineering
The Ohio State University
Columbus, OH 43210
mbelkin@cse.ohio-state.edu

**Partha Niyogi**
Department of Computer Science
University of Chicago
Chicago IL 60637
niyogi@cs.uchicago.edu

## Abstract

One of the intuitions underlying many graph-based methods for clustering and semi-supervised learning, is that class or cluster boundaries pass through areas of low probability density. In this paper we provide some formal analysis of that notion for a probability distribution. We introduce a notion of weighted boundary volume, which measures the length of the class/cluster boundary weighted by the density of the underlying probability distribution. We show that sizes of the cuts of certain commonly used data adjacency graphs converge to this continuous *weighted volume* of the boundary.

**keywords:** Clustering, Semi-Supervised Learning

## 1 Introduction

Consider the probability distribution with density $p(x)$ depicted in Fig. 1, where darker color denotes higher probability density. Asked to *cluster* this probability distribution, we would probably separate it into two roughly Gaussian bumps as shown in the left panel.

Same intuition applies to semi-supervised learning. Asked to point out more likely groups of data of the same type, we would be inclined to believe that these two bumps contain data points with the same labels. On the other hand, the class boundary shown in the right panel seems rather less likely. One way to state this basic intuition is the Low Density Separation assumption [5], saying that the class/cluster boundary tends to pass through regions of low density.

In this paper we propose a formal measure on the complexity of the boundary, which intuitively corresponds to the Low Density Separation assumption. We will show that given a class boundary, this measure can be computed from a finite sample from the probability distribution. Moreover, we show this is done by computing the size of a cut for a partition of a certain standard adjacency graph, defined on that sample, and point out some interesting connections to spectral clustering.

To fix our intuitions, let us consider the question of what makes the cut in the left panel more intuitively acceptable than the cut in the right. Two features of the left cut make it more pleasing: the cut is shorter in length and the cut

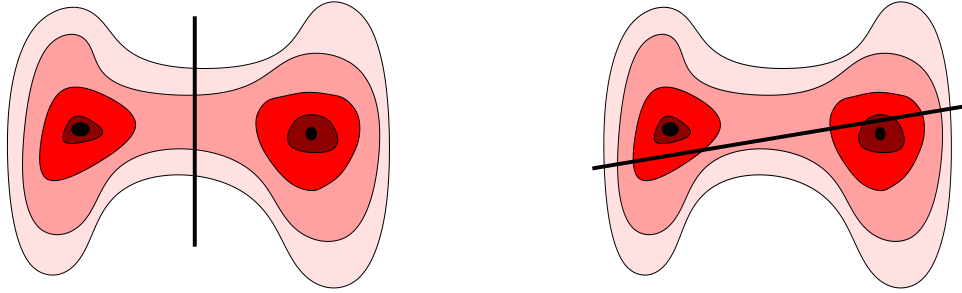

Figure 1: A likely cut and a less likely cut.

passes through a low-density area. Note that a very jagged cut through a low-density area or a short cut through the middle of a high-density bump would be unsatisfactory. It will therefore appear reasonable to take the length of the cut as a measure of its complexity but weight it depending on the density of the probability distribution $p$ through which it passes. In other words, we propose the *weighted length* of the boundary, represented by the contour integral along the boundary $\int_{\text{cut}} p(s)ds$ to measure the complexity of a cut. It is clear that the boundary in the left panel has a considerably lower weighted length than the boundary in the right panel of our Fig. 1.

To formalize this notion further consider a (marginal) probability distribution with density $p(x)$ supported on some domain or manifold $M$. This domain is partitioned in two disjoint clusters/parts. Assuming that the boundary $S$ is a smooth hypersurface we define the *weighted volume of the cut* to be $\int_S p(s)ds$. Note that just as in the example above, the integral is taken over the surface of the boundary.

We will show how this quantity can be approximated given empirical data and establish connections with some popular graph-based methods.

## 2 Connections and related work

### 2.1 Spectral Clustering

Over the last two decades there has been considerable interest in various spectral clustering techniques (see, e.g., [6] for an overview). The idea of spectral clustering can be expressed very simply. Given a graph, we would often like to construct a balanced partitioning of the vertex set, i.e. a partitioning such which minimizes the number (or total weight) of edges across the cut. This is generally an NP-hard optimization problem. It turns out, however, that a simple real-valued relaxation can be used to reduce it to standard linear algebra, typically to finding eigenvectors of a certain *graph Laplacian*. We note that the quality of partition is usually measured in terms of the corresponding *cut size*.

A critical question, when this notion is applied to general purpose clustering in the context of machine learning is how to construct the graph given data points. A typical choice here is the Gaussian weights (e.g., [14]). To summarize, a graph is obtained from a point cloud, using Gaussian or other weights, and partitioned using spectral clustering or a different algorithm, which attempts to approximate the smallest (balanced) cut.

We note that while the intuition is that spectral clustering is an approximation to the minimum cut, and is closely related to random walks and diffusions on graphs and the underlying probability distributions ([13, 12]), existing results on convergence of spectral clustering ([11]) do not provide a formal interpretation of the limiting partition or connect it to the size of the resulting cut.

### 2.2 Graph-based semi-supervised learning

Similarly to spectral clustering, graph-based semi-supervised learning constructs a graph from the data. In contrast to clustering, however, some of the data is labeled. The problem is typically to either label the unlabeled points (transduction) or, more generally, to build a classifier defined on the whole space. This may be done trying to find the

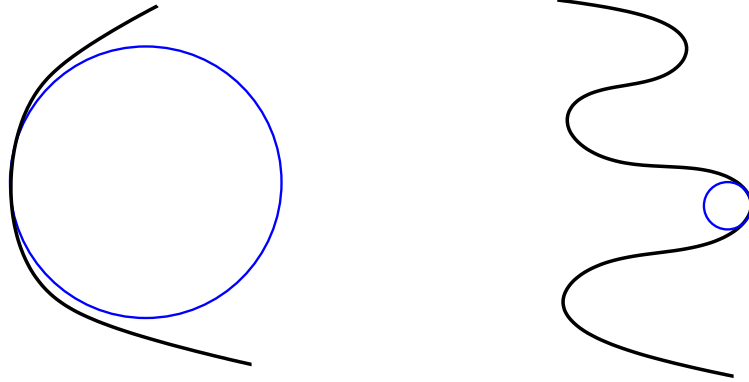

Figure 2: Curves of small and high condition number respectively

minimum cut, which respects the labels of the data directly ([3]), or, using graph Laplacian as a penalty functional (e.g.,[15, 1]).

One of the important intuitions of semi-supervised learning is the *cluster assumption*(e.g., [4]) or, more specifically, the *low density separation assumption* suggested in [5], which states that the class boundary passes through a low density region. We argue that this intuition needs to slightly modified by suggesting that cutting through a high density region may be acceptable as long as the length of the cut is very short. For example imagine two high-density round clusters connected by a very thin high-density thread. Cutting the thread is appropriate as long as the width of the thread is much smaller than the radii of the clusters.

### 2.3 Convergence of Manifold Laplacians

Another closely related line of research is the connections between point-cloud graph Laplacians and Laplace-Beltrami operators on manifolds, which have been explored recently in [9, 2, 10]. A typical result in that setting shows that for a fixed function $f$ and points sampled from a probability distribution on a manifold or a domain, the graph Laplacian applied to $f$ converges to the manifold Laplace-Beltrami operator $\Delta_{\mathcal{M}} f$. We note that the results of those papers cannot be directly applied in our situation as for us $f$ is the indicator function of a subset and is not differentiable. Even more importantly, this paper establishes an explicit connection between the point-cloud Laplacian applied to such characteristic functions (weighted graph cuts) and a geometric quantity, which is the weighted volume of the cut boundary. This geometric connection does not easily follow from results of those papers and the techniques used in the proof of our Theorem 3 are significantly different.

## 3 Summary of the Main Results

Let $p$ be a probability density function on a domain $M \subseteq \mathbb{R}^d$.

Let $S$ be a smooth hypersurface that separates $M$ into two parts, $S_1$ and $S_2$. The smoothness of $S$ will be quantified by a *condition number* $1/\tau$, where $\tau$ is the radius of the largest ball that can be placed tangent to the manifold at any point, intersecting the manifold at only one point. It bounds the curvature of the manifold.

**Definition 1** *Let $K_t(x, y)$ be the heat kernel in $\mathbb{R}^d$ given by*

$$K_t(x, y) := \frac{1}{(4\pi t)^{d/2}} \, e^{-\|x-y\|^2/4t}.$$

*Let $M_t := K_t(x, x) = \frac{1}{(4\pi t)^{d/2}}$.*

Let $X := \{x_1, \ldots, x_N\}$ be a set of $N$ points chosen independently at random from $p$. Consider the complete graph whose vertices are associated with the points in $X$, and where the weight of the edge between $x_i$ and $x_j$, $i \neq j$ is given

by

$$W_{ij} = K_t(x_i, x_j)$$

Let $W$ be the weight matrix. Let $X_1 = X \cap S_1$ and $X_2 = X \cap S_2$ be the data point which land in $S_1$ and $S_2$ respectively. Let $D$ be the diagonal matrix whose entries are row sums of $W$ (degrees of the corresponding vertices)

$$D_{ii} = \sum_j W_{ij}$$

The *normalized Laplacian* associated to the data $X$ (and parameter $t$) is the matrix $L(t, X) := I - D^{-1/2}WD^{-1/2}$.

Let $f = (f_1, \ldots, f_N)$ be the indicator vector for $X_1$:

$$f_i = \begin{cases} 1 & \text{if } x_i \in X_1 \\ 0 & \text{otherwise} \end{cases}$$

There are two quantities of interest:

1. $\int_S p(s)ds$, which measures the quality of the partition $S$ in accordance with the *weighted volume of the boundary*.
2. $f^T L f$, which measures the quality of the empirical partition in terms of its *cut size*.

Our main Theorem shows that after an appropriate scaling, the empirical cut size converges to the volume of the boundary.

**Theorem 1** *Let the number of points, $|X| = N$ tend to infinity and $\{t_N\}_0^\infty$, be a sequence of values of $t$ that tend to zero such that $t_N > \frac{1}{N^{\frac{1}{2d+2}}}$. Then with probability 1,*

$$\lim \frac{\sqrt{\pi}}{N\sqrt{t}} f^T L(t_N, X) f = \int_S p(s)ds$$

*Further, for any $\delta \in (0, 1)$ and any $\epsilon \in (0, 1/2)$, there exists a positive constant $C$ and an integer $N_0$(depending on $\epsilon$, $\delta$ and certain generic invariants of $p$ and $S$) such that with probability $1 - \delta$, $(\forall N > N_0)$,*

$$\left| \frac{\sqrt{\pi}}{N\sqrt{t}} f^T L(t_N, X) f - \int_S p(s)ds \right| < Ct_N^\epsilon.$$

This theorem is proved by first relating the empirical quantity $\frac{\sqrt{\pi}}{N\sqrt{t}} f^T L(t_N, X) f$ to a heat flow across the relevant cut (on the continuous domain), and then relating the heat flow to the measure of the cut. In order to state these results, we need the following notation.

**Definition 2** *Let*

$$\psi_t(x) = \frac{p(x)}{\sqrt{\int_M K_t(x, z)p(z)dz}}.$$

*Let*

$$\beta(t, X) := \frac{\sqrt{\pi}}{N\sqrt{t}} f^T L(t, X) f$$

*and*

$$\alpha(t) := \sqrt{\frac{\pi}{t}} \int_{S_1} \int_{S_2} K_t(x, y)\psi_t(x)\psi_t(y)dxdy.$$

Where $t$ and $X$ are clear from context, we shall abbreviate $\beta(t, X)$ to $\beta$ and $\alpha(t)$ to $\alpha$. In theorem 2 we show that for a fixed $t$, as the number of points $|X| = N$ tends to infinity, with probability 1, $\beta(t, X)$ tends to $\alpha(t)$.

In theorem 3 we show that $\alpha(t)$ can be made arbitrarily close to the weighted volume of the boundary by making $t$ tend to 0.

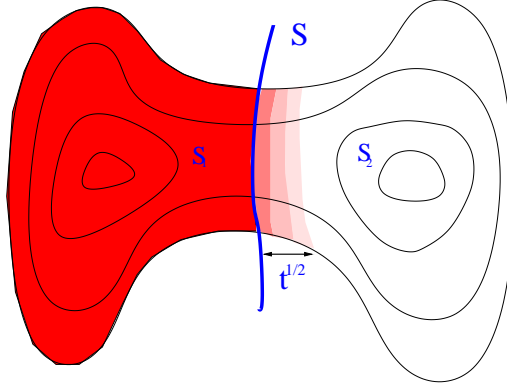

Figure 3: Heat flow $\alpha$ tends to $\int_S p(s)ds$

**Theorem 2** *Let $0 < \mu < 1$. Let $u := 1/\sqrt{t^{2d+1}N^{1-\mu}}$. Then, there exist positive constants $C_1, C_2$ depending only on $p$ and $S$ such that with probability greater than $1 - \exp\left(-C_1 N^\mu\right)$*

$$|\beta(t, X) - \alpha(t)| < C_2 \left(1 + t^{\frac{d+1}{2}}\right) u\alpha(t). \tag{1}$$

**Theorem 3** *For any $\epsilon \in (0, \frac{1}{2})$, there exists a constant $C$ such that for all $t$ such that $0 < t < \tau(2d)^{-\frac{e}{e-1}}$,*

$$\left|\sqrt{\frac{\pi}{t}} \int_{S_2} \int_{S_1} K_t(x, y)\psi_t(x)\psi_t(y)dxdy - \int_S p(s)ds\right| < C\, t^\epsilon. \tag{2}$$

By letting $N \to \infty$ and $t_N \to 0$ at suitable rates and putting together theorems 2 and 3, we obtain the following theorem:

**Theorem 4** *Let the number of random data points $N \to \infty$, and $t_N \to 0$, at rates so that $u := 1/\sqrt{t^{2d+1}N^{1-\mu}} \to 0$. Then, for any $\epsilon \in (0, 1/2)$, there exist positive constants $C_1, C_2$ depending only on $p$ and $S$, such that for any $N > 1$ with probability greater than $1 - \exp\left(-C_1(N^\mu)\right)$,*

$$\left|\beta\left(t_N, X\right) - \int_S p(s)ds\right| < C_2\left(t^\epsilon + u\right) \tag{3}$$

## 4   Outline of Proofs

Theorem 1 is a corollary of Theorem 4, obtained by setting $u$ to be $t^\epsilon$, and setting $\mu$ to $\frac{1-2\epsilon}{2d+2}$. $N_0$ is chosen to be a large enough integer so that an application of the union bound on all $N > N_0$, still gives us a probability $\sum_{N > N_0} \exp\left(-C_1(N^\mu)\right) < \delta$, of the rate of convergence being worse than stated in Theorem 1.

Theorem 4 is a direct consequence of Theorem 2 and Theorem 3.

**Theorem 2:**

We prove theorem 2 using a generalization of McDiarmid's inequality from [7, 8]. McDiarmid's inequality asserts that a function of a large number of independent random variables, that is not very influenced by the value of any one of these, takes a value close to its mean. In the generalization that we use, it is permitted that over a bad set that has a small probability mass, the function is highly influenced by some of the random variables. In our setting, it can be shown that our measure of a cut, $f^T L f$ is such a function of the independent random points in $X$, and so the result is applicable. There is another step involved, since the mean of $f^T L f$ is not $\alpha$, the quantity to which we wish to prove convergence. Therefore we need to prove that the mean $E[\frac{\sqrt{\pi}}{N\sqrt{t}} f^T L(t, X)f]$ tends to $\alpha(t)$ as $N$ tends to infinity. Now,

$$\frac{\sqrt{\pi}}{N\sqrt{t}} f^T L(t, X)f = 1/N\sqrt{\pi/t} \sum_{x \in X_1} \sum_{y \in X_2} \frac{K_t(x, y)}{\{(\sum_{z \neq x} K_t(x, z))(\sum_{z \neq y} K_t(y, z))\}^{1/2}}.$$

If, instead, we had in the denominator of the right side

$$\sqrt{\int\limits_M p(z)K_t(x,z)dz \int\limits_M p(z)K_t(y,z)dz,}$$

using the linearity of Expectation,

$$E\left[\frac{1}{N(N-1)}\sqrt{\pi/t}\sum_{x\in X_1}\sum_{y\in X_2}\frac{K_t(x,y)}{\sqrt{\left(\int\limits_M p(z)K_t(x,z)dz\right)\left(\int\limits_M p(z)K_t(y,z)dz\right)}}\right] = \alpha(t).$$

Using Chernoff bounds, we can show that with high probability, for all $x \in X$,

$$\frac{\sum_{z\neq x}K_t(x,z)}{N-1} \approx \int\limits_M p(z)K_t(x,z)dz.$$

Putting the last two facts together and using the Generalization of McDiarmid's inequality from [7, 8], the result follows. Since the exact details require fairly technical calculations, we leave them to the Journal version.

**Theorem 3:**

The quantity

$$\alpha := \sqrt{\frac{\pi}{t}}\int\limits_{S_1}\int\limits_{S_2}K_t(x,y)\psi_t(x)\psi_t(y)dxdy$$

is similar to the heat that would flow in time $t$ from one part to another if the first were heated proportional to $p$. Intuitively, the heat that would flow from one part to the other in a small interval ought to be related to the volume of the boundary between these two parts, which in our setting is $\int_S p(s)ds$. To prove this relationship, we bound $\alpha$ both above and below in terms of the weighted volume and condition number of the boundary. These bounds are obtained by making comparisons with the "worst case", given condition number $\frac{1}{\tau}$, which is when $S$ is a sphere of radius $\tau$. In order to obtain a lower bound on $\alpha$, we observe that if $B_2$ is the nearest ball of radius $\tau$ contained in $S_1$ to a point $P$ in $S_2$ that is within $\tau$ of $S_1$,

$$\int\limits_{S_1}K_t(x,P)\psi_t(x)\psi_t(P)dx \geq \int\limits_{B_2}K_t(x,P)\psi_t(x)\psi_t(P)dx,$$

as in Figure 4.

Similarly, to obtain an upper bound on $\alpha$, we observe that if $B_1$ is a ball or radius $\tau$ in $S_2$, tangent to $B_2$ at the point of $S$ nearest to $P$,

$$\int\limits_{S_1}K_t(x,P)\psi_t(x)\psi_t(P)dx \leq \int\limits_{B_1^c}K_t(x,P)\psi_t(x)\psi_t(P)dx.$$

We now indicate how a lower bound is obtained for

$$\int\limits_{B_2}K_t(x,P)\psi_t(x)\psi_t(P)dx.$$

A key observation is that for $R = \sqrt{2dt\ln(1/t)}$, $\int\limits_{\|x-P\|>R}K_t(x,P)dx \ll 1$. For this reason, only the portions of $B_2$ near $P$ contribute to the the integral

$$\int\limits_{B_2}K_t(x,P)\psi_t(x)\psi_t(P)dx.$$

It turns out that a good lower bound can be obtained by considering the integral over $H_2$ instead, where $H_2$ is a halfspace whose boundary is at a distance $\tau - \frac{R^2}{2\tau}$ from the center as in figure 4.

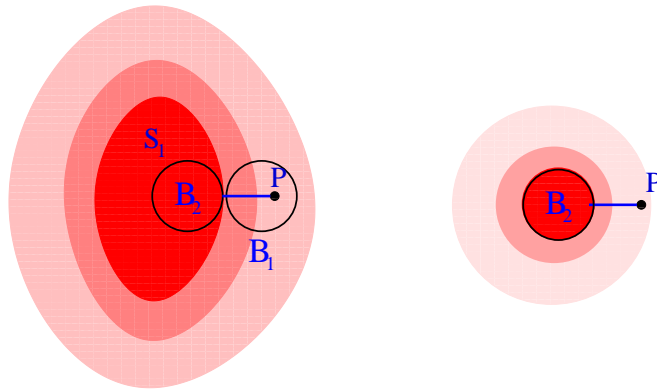

Figure 4: The density of heat diffusing to point $P$ from $S_1$ in the left panel is less or equal to the density of heat diffusing to $P$ from $B_2$ in the right panel.

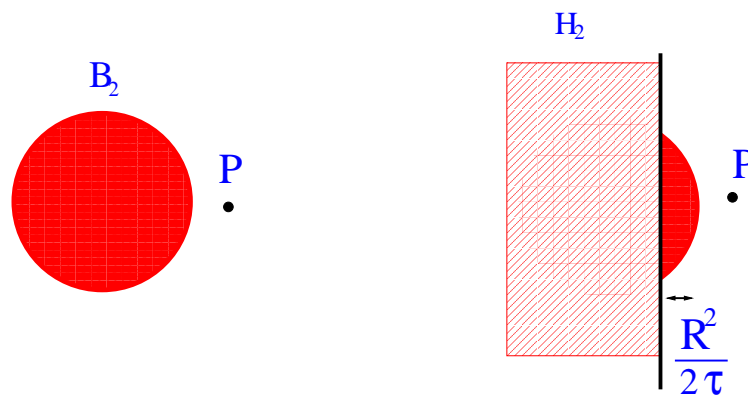

Figure 5: The density of heat received by point P from $B_2$ in the left panel can be approximated by the density of heat received by P from the halfspace $H_2$ in the right panel.

An upper bound for

$$\int\limits_{B_1^c} K_t(x, P)\psi_t(x)\psi_t(P)dx$$

is obtained along similar lines.

The details of this proof will be presented in the Journal version.

## 5 Conclusion

In this paper we take a step towards a probabilistic analysis of graph based methods for clustering. The nodes of the graph are identified with data points drawn at random from an underlying probability distribution on a continuous domain. For a fixed partition we show that the cut-size of the graph partition converges to the weighted volume of the boundary separating the two regions of the domain. The rates of this convergence are analyzed. If one is able to generalize our result uniformly over all partitions, this allows us to relate ideas around graph based partitioning to ideas surrounding Low Density Separation. The most important future direction would be to achieve similar results uniformly over balanced partitions.

## References

[1] M. Belkin and P. Niyogi (2004)."Semi-supervised Learning on Riemannian Manifolds." In *Machine Learning* 56, Special Issue on Clustering, 209-239.

[2] M. Belkin and P. Niyogi. "Toward a theoretical foundation for Laplacian-based manifold methods." COLT 2005.

[3] A. Blum and S. Chawla, "Learning from labeled and unlabeled data using graph mincuts", ICML 2001.

[4] O.Chapelle, J. Weston, B. Scholkopf, "Cluster kernels for semi-supervised learning", NIPS 2002.

[5] O. Chapelle and A. Zien, "Semi-supervised Classification by Low Density Separation", AISTATS 2005.

[6] Chris Ding, Spectral Clustering, ICML 2004 Tutorial.

[7] Samuel Kutin, Partha Niyogi, "Almost-everywhere Algorithmic Stability and Generalization Error.", UAI 2002, 275-282

[8] S. Kutin, TR-2002-04, "Extensions to McDiarmid's inequality when differences are bounded with high probability." Technical report TR-2002-04 at the Department of Computer Science, University of Chicago.

[9] S. Lafon, *Diffusion Maps and Geodesic Harmonics*, Ph. D. Thesis, Yale University, 2004.

[10] M. Hein, J.-Y. Audibert, U. von Luxburg, *From Graphs to Manifolds – Weak and Strong Pointwise Consistency of Graph Laplacians*, COLT 2005.

[11] U. von Luxburg, M. Belkin, O. Bousquet, *Consistency of Spectral Clustering*, Max Planck Institute for Biological Cybernetics Technical Report TR 134, 2004.

[12] M. Meila and J. Shi. *A Random Walks View of Spectral Segmentation*, NIPS 2001.

[13] B. Nadler, S. Lafon, R. R. Coifman,and I. G. Kevrekidis. *Diffusion Maps, Spectral Clustering and Eigenfunctions of Fokker-Planck Operators*, NIPS 2006.

[14] J. Shi and J. Malik. "Normalized cuts and image segmentation."

[15] X. Zhu, J. Lafferty and Z. Ghahramani, *Semi-supervised learning using Gaussian fields and harmonic functions*, ICML 2003.

